# A novel family of non-parametric cumulative based divergences for point processes

**Sohan Seth**
University of Florida

**Il "Memming" Park**
University of Texas at Austin

**Austin J. Brockmeier**
University of Florida

**Mulugeta Semework**
SUNY Downstate Medical Center

**John Choi, Joseph T. Francis**
SUNY Downstate Medical Center & NYU-Poly

**José C. Príncipe**
University of Florida

## Abstract

Hypothesis testing on point processes has several applications such as model fitting, plasticity detection, and non-stationarity detection. Standard tools for hypothesis testing include tests on mean firing rate and time varying rate function. However, these statistics do not fully describe a point process, and therefore, the conclusions drawn by these tests can be misleading. In this paper, we introduce a family of non-parametric divergence measures for hypothesis testing. A divergence measure compares the full probability structure and, therefore, leads to a more robust test of hypothesis. We extend the traditional Kolmogorov–Smirnov and Cramér–von-Mises tests to the space of spike trains via stratification, and show that these statistics can be consistently estimated from data without any free parameter. We demonstrate an application of the proposed divergences as a cost function to find optimally matched point processes.

## 1 Introduction

Neurons communicate mostly through noisy *sequences of action potentials*, also known as *spike trains*. A *point process* captures the stochastic properties of such sequences of events [1]. Many *neuroscience* problems such as model fitting (goodness-of-fit), plasticity detection, change point detection, non-stationarity detection, and neural code analysis can be formulated as statistical inference on point processes [2, 3]. To avoid the complication of dealing with spike train observations, neuroscientists often use summarizing statistics such as mean firing rate to compare two point processes. However, this approach implicitly assumes a model for the underlying point process, and therefore, the choice of the summarizing statistic fundamentally restricts the validity of the inference procedure.

One alternative to mean firing rate is to use the distance between the inhomogeneous rate functions, i.e. $\int |\lambda_1(t) - \lambda_2(t)|\, \mathrm{d}t$, as a test statistic, which is sensitive to the temporal fluctuation of the means of the point processes. In general the rate function does not fully specify a point process, and therefore, ambiguity occurs when two distinct point processes have the same rate function. Although physiologically meaningful change is often accompanied by the change in rate, there has been evidence that the higher order statistics can change without a corresponding change of rate [4, 5]. Therefore, statistical tools that capture higher order statistics, such as *divergences*, can improve the state-of-the-art hypothesis testing framework for spike train observations, and may encourage new scientific discoveries.

In this paper, we present a novel family of divergence measures between two point processes. Unlike firing rate function based measures, a divergence measure is zero *if and only if* the two point processes are identical. Applying a divergence measure for hypothesis testing is, therefore, more appropriate in a statistical sense. We show that the proposed measures can be estimated from data without any assumption on the underlying probability structure. However, a distribution-free (non-parametric) approach often suffers from having free parameters, e.g. choice of kernel in non-parametric density estimation, and these free parameters often need to be chosen using computationally expensive methods such as cross validation [6]. We show that the proposed measures can be consistently estimated in a *parameter free* manner, making them particularly useful in practice.

One of the difficulties of dealing with continuous-time point process is the lack of well structured space on which the corresponding probability laws can be described. In this paper we follow a rather unconventional approach for describing the point process by a direct sum of Euclidean spaces of varying dimensionality, and show that the proposed divergence measures can be expressed in terms of cumulative distribution functions (CDFs) in these disjoint spaces. To be specific, we represent the point process by the probability of having a finite number of spikes and the probability of spike times given that number of spikes, and since these time values are reals, we can represent them in a Euclidean space using a CDF. We follow this particular approach since, first, CDFs can be easily estimated consistently using empirical CDFs without any free parameter, and second, standard tests on CDFs such as Kolmogorov–Smirnov (K-S) test [7] and Cramér–von-Mises (C-M) test [8] are well studied in the literature. Our work extends the conventional K-S test and C-M test on the real line to the space of spike trains.

The rest of the paper is organized as follows; in section 2 we introduce the measure space where the point process is defined as probability measures, in section 3 and section 4 we introduce the extended K-S and C-M divergences, and derive their respective estimators. Here we also prove the consistency of the proposed estimators. In section 5, we compare various point process statistics in a hypothesis testing framework. In section 6 we show an application of the proposed measures in selecting the optimal stimulus parameter. In section 7, we conclude the paper with some relevant discussion and future work guidelines.

## 2  Basic point process

We define a point process to be a probability measure over all possible spike trains. Let $\Omega$ be the set of all finite spike trains, that is, each $\omega \in \Omega$ can be represented by a finite set of action potential timings $\omega = \{t_1 \leq t_2 \leq \ldots \leq t_n\} \in \mathbb{R}^n$ where $n$ is the number of spikes. Let $\Omega_0, \Omega_1, \cdots$ denote the partitions of $\Omega$ such that $\Omega_n$ contains all possible spike trains with exactly $n$ events (spikes), hence $\Omega_n = \mathbb{R}^n$. Note that $\Omega = \bigcup_{n=0}^{\infty} \Omega_n$ is a disjoint union, and that $\Omega_0$ has only one element representing the empty spike train (no action potential). See Figure 1 for an illustration.

Define a $\sigma$-algebra on $\Omega$ by the $\sigma$-algebra generated by the union of Borel sets defined on the Euclidean spaces; $\mathcal{F} = \sigma\left(\bigcup_{n=0}^{\infty} \mathcal{B}\left(\Omega_n\right)\right)$. Note that any measurable set $A \in \mathcal{F}$ can be partitioned into $\{A_n = A \cap \Omega_n\}_{n=0}^{\infty}$, such that each $A_n$ is measurable in corresponding measurable space $(\Omega_n, \mathcal{B}(\Omega_n))$. Here $A$ denotes a collection of spike trains involving varying number of action potentials and corresponding action potential timings, whereas $A_n$ denotes a subset of these spike trains involving only $n$ action potentials each.

A (finite) point process is defined as a probability measure $P$ on the measurable space $(\Omega, \mathcal{F})$ [1]. Let $P$ and $Q$ be two probability measures on $(\Omega, \mathcal{F})$, then we are interested in finding the divergence $d(P, Q)$ between $P$ and $Q$, where a divergence measure is characterized by $d(P, Q) \geq 0$ and $d(P, Q) = 0 \iff P = Q$.

## 3  Extended K-S divergence

A Kolmogorov-Smirnov (K-S) type divergence between $P$ and $Q$ can be derived from the $L_1$ distance between the probability measures, following the equivalent representation,

$$d_1(P, Q) = \int_{\Omega} d\,|P - Q| \geq \sup_{A \in \mathcal{F}} |P(A) - Q(A)|. \tag{1}$$

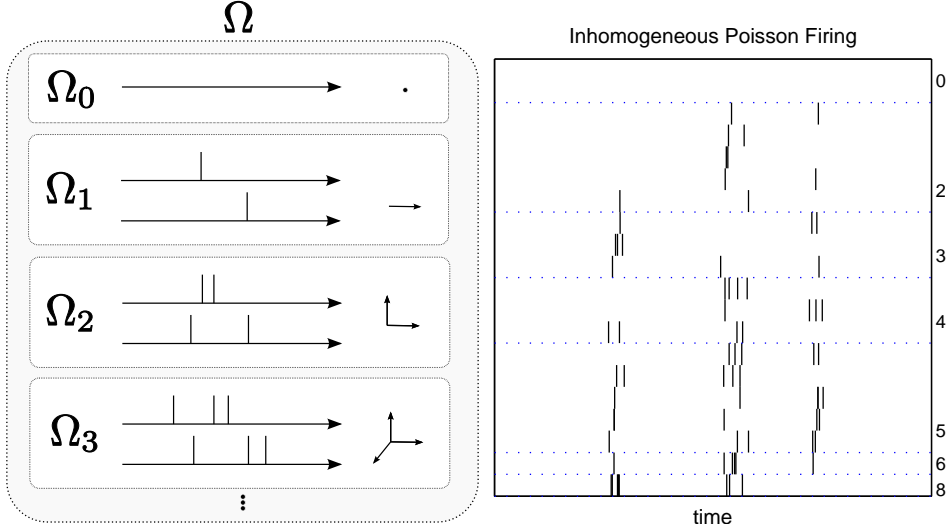

Figure 1: (Left) Illustration of how the point process space is stratified. (Right) Example of spike trains stratified by their respective spike count.

Since (1) is difficult and perhaps impossible to estimate directly without a model, our strategy is to use the stratified spaces $(\Omega_0, \Omega_1, \ldots)$ defined in the previous section, and take the supremum only in the corresponding conditioned probability measures. Let $\mathcal{F}_i = \mathcal{F} \cap \Omega_i := \{F \cap \Omega_i | F \in \mathcal{F}\}$. Since $\cup_i \mathcal{F}_i \subset \mathcal{F}$,

$$d_1(P, Q) \geq \sum_{n \in \mathbb{N}} \sup_{A \in \mathcal{F}_n} |P(A) - Q(A)| = \sum_{n \in \mathbb{N}} \sup_{A \in \mathcal{F}_n} |P(\Omega_n)P(A|\Omega_n) - Q(\Omega_n)Q(A|\Omega_n)| .$$

Since each $\Omega_n$ is a Euclidean space, we can induce the traditional K-S test statistic by further reducing the search space to $\tilde{\mathcal{F}}_n = \{\times_i(-\infty, t_i] | t = (t_1, \ldots, t_n) \in \mathbb{R}^n\}$. This results in the following inequality,

$$\sup_{A \in \mathcal{F}_n} |P(A) - Q(A)| \geq \sup_{A \in \tilde{\mathcal{F}}_n} |P(A) - Q(A)| = \sup_{t \in \mathbb{R}^n} \left| F_P^{(n)}(t) - F_Q^{(n)}(t) \right|, \quad (2)$$

where $F_P^{(n)}(t) = P[T_1 \leq t_1 \wedge \ldots \wedge T_n \leq t_n]$ is the cumulative distribution function (CDF) corresponding to the probability measure $P$ in $\Omega_n$. Hence, we define the K-S divergence as

$$d_{KS}(P, Q) = \sum_{n \in \mathbb{N}} \sup_{t \in \mathbb{R}^n} \left| P(\Omega_n)F_P^{(n)}(t) - Q(\Omega_n)F_Q^{(n)}(t) \right|. \quad (3)$$

Given a finite number of samples $X = \{x_i\}_{i=1}^{N_P}$ and $Y = \{y_j\}_{j=1}^{N_Q}$ from $P$ and $Q$ respectively, we have the following estimator for equation (3).

$$\hat{d}_{KS}(P, Q) = \sum_{n \in \mathbb{N}} \sup_{t \in \mathbb{R}^n} \left| \hat{P}(\Omega_n)\hat{F}_P^{(n)}(t) - \hat{Q}(\Omega_n)\hat{F}_Q^{(n)}(t) \right|$$

$$= \sum_{n \in \mathbb{N}} \sup_{t \in X_n \cup Y_n} \left| \hat{P}(\Omega_n)\hat{F}_P^{(n)}(t) - \hat{Q}(\Omega_n)\hat{F}_Q^{(n)}(t) \right|, \quad (4)$$

where $X_n = X \cap \Omega_n$, and $\hat{P}$ and $\hat{F}_P$ are the empirical probability and empirical CDF, respectively. Notice that we only search the supremum over the locations of the realizations $X_n \cup Y_n$ and not the whole $\mathbb{R}^n$, since the empirical CDF difference $\left| \hat{P}(\Omega_n)\hat{F}_P^{(n)}(t) - \hat{Q}(\Omega_n)\hat{F}_Q^{(n)}(t) \right|$ only changes values at those locations.

**Theorem 1** ($d_{KS}$ is a divergence)**.**

$$d_1(P, Q) \geq d_{KS}(P, Q) \geq 0 \quad (5)$$
$$d_{KS}(P, Q) = 0 \iff P = Q \quad (6)$$

*Proof.* The first property and the $\Leftarrow$ proof for the second property are trivial. From the definition of $d_{KS}$ and properties of CDF, $d_{KS}(P, Q) = 0$ implies that $P(\Omega_n) = Q(\Omega_n)$ and $F_P^{(n)} = F_Q^{(n)}$ for all $n \in \mathbb{N}$. Given probability measures for each $(\Omega_n, \mathcal{F}_n)$ denoted as $P_n$ and $Q_n$, there exist corresponding unique extended measures $P$ and $Q$ for $(\Omega, \mathcal{F})$ such that their restrictions to $(\Omega_n, \mathcal{F}_n)$ coincide with $P_n$ and $Q_n$, hence $P = Q$. $\qquad\square$

**Theorem 2** (Consistency of K-S divergence estimator). *As the sample size approaches infinity,*

$$\left| d_{KS} - \hat{d}_{KS} \right| \xrightarrow{a.u.} 0 \tag{7}$$

*Proof.* Note that $\left| \sum \sup \cdot - \sum \sup \cdot \right| \leq \sum |\sup \cdot - \sup \cdot|$. Due to the triangle inequality of the supremum norm,

$$\left| \sup_{t \in \mathbb{R}^n} \left| P(\Omega_n) F_P^{(n)}(t) - Q(\Omega_n) F_Q^{(n)}(t) \right| - \sup_{t \in \mathbb{R}^n} \left| \hat{P}(\Omega_n) \hat{F}_P^{(n)}(t) - \hat{Q}(\Omega_n) \hat{F}_Q^{(n)}(t) \right| \right|$$

$$\leq \sup_{t \in \mathbb{R}^n} \left| \left| P(\Omega_n) F_P^{(n)}(t) - Q(\Omega_n) F_Q^{(n)}(t) \right| - \left| \hat{P}(\Omega_n) \hat{F}_P^{(n)}(t) - \hat{Q}(\Omega_n) \hat{F}_Q^{(n)}(t) \right| \right|.$$

Again, using the triangle inequality we can show the following:

$$\left| \left| P(\Omega_n) F_P^{(n)}(t) - Q(\Omega_n) F_Q^{(n)}(t) \right| - \left| \hat{P}(\Omega_n) \hat{F}_P^{(n)}(t) - \hat{Q}(\Omega_n) \hat{F}_Q^{(n)}(t) \right| \right|$$

$$\leq \left| P(\Omega_n) F_P^{(n)}(t) - Q(\Omega_n) F_Q^{(n)}(t) - \hat{P}(\Omega_n) \hat{F}_P^{(n)}(t) + \hat{Q}(\Omega_n) \hat{F}_Q^{(n)}(t) \right|$$

$$= \left| P(\Omega_n) F_P^{(n)}(t) - P(\Omega_n) \hat{F}_P^{(n)}(t) - Q(\Omega_n) F_Q^{(n)}(t) + Q(\Omega_n) \hat{F}_Q^{(n)}(t) \right.$$

$$\left. + P(\Omega_n) \hat{F}_P^{(n)}(t) - \hat{P}(\Omega_n) \hat{F}_P^{(n)}(t) + \hat{Q}(\Omega_n) \hat{F}_Q^{(n)}(t) - Q(\Omega_n) \hat{F}_Q^{(n)}(t) \right|$$

$$\leq P(\Omega_n) \left| F_P^{(n)}(t) - \hat{F}_P^{(n)}(t) \right| + Q(\Omega_n) \left| F_Q^{(n)}(t) - \hat{F}_Q^{(n)}(t) \right|$$

$$+ \hat{F}_P^{(n)}(t) \left| P(\Omega_n) - \hat{P}(\Omega_n) \right| + \hat{F}_Q^{(n)}(t) \left| Q(\Omega_n) - \hat{Q}(\Omega_n) \right|.$$

Then the theorem follows from the Glivenko-Cantelli theorem, and $\hat{P}, \hat{Q} \xrightarrow{a.s.} P, Q$. $\qquad\square$

Notice that the inequality in (2) can be made stricter by considering the supremum over not just the product of the segments $(-\infty, t_i]$ but over the all $2^n - 1$ possible products of the segments $(-\infty, t_i]$ and $[t_i, \infty)$ in $n$ dimensions [7]. However, the latter approach is computationally more expensive, and therefore, in this paper we only explore the former approach.

## 4 Extended C-M divergence

We can extend equation (3) to derive a Cramér–von-Mises (C-M) type divergence for point processes. Let $\mu = P + Q/2$, then $P, Q$ are absolutely continuous with respect to $\mu$. Note that, $F_P^{(n)}, F_Q^{(n)} \in L_2(\Omega_n, \mu|_n)$ where $|_n$ denotes the restriction on $\Omega_n$, i.e. the CDFs are $L_2$ integrable, since they are bounded. Analogous to the relation between K-S test and C-M test, we would like to use the integrated squared deviation statistics in place of the maximal deviation statistic. By integrating over the probability measure $\mu$ instead of the supremum operation, and using $L_2$ instead of $L_\infty$ distance, we define

$$d_{CM}(P, Q) = \sum_{n \in \mathbb{N}} \int_{\mathbb{R}^n} \left( P(\Omega_n) F_P^{(n)}(t) - Q(\Omega_n) F_Q^{(n)}(t) \right)^2 \mathrm{d}\mu|_n(t). \tag{8}$$

This can be seen as a direct extension of the C-M criterion. The corresponding estimator can be derived using the strong law of large numbers,

$$\hat{d}_{CM}(P, Q) = \sum_{n \in \mathbb{N}} \left[ \frac{1}{2} \sum_i \left( \hat{P}(\Omega_n) \hat{F}_P^{(n)}(x_i^{(n)}) - \hat{Q}(\Omega_n) \hat{F}_Q^{(n)}(x_i^{(n)}) \right)^2 \right.$$

$$\left. + \frac{1}{2} \sum_i \left( \hat{P}(\Omega_n) \hat{F}_P^{(n)}(y_i^{(n)}) - \hat{Q}(\Omega_n) \hat{F}_Q^{(n)}(y_i^{(n)}) \right)^2 \right]. \tag{9}$$

**Theorem 3** ($d_{CM}$ is a divergence). *For $P$ and $Q$ with square integrable CDFs,*

$$d_{CM}(P,Q) \geq 0 \tag{10}$$
$$d_{CM}(P,Q) = 0 \iff P = Q. \tag{11}$$

*Proof.* Similar to theorem 1. □

**Theorem 4** (Consistency of C-M divergence estimator). *As the sample size approaches infinity,*

$$\left| d_{CM} - \hat{d}_{CM} \right| \xrightarrow{a.u.} 0 \tag{12}$$

*Proof.* Similar to (7), we find an upper bound and show that the bound uniformly converges to zero. To simplify the notation, we define $g_n(x) = P(\Omega_n)F_P^{(n)}(x) - Q(\Omega_n)F_Q^{(n)}(x)$, and $\hat{g}_n(x) = \hat{P}(\Omega_n)\hat{F}_P^{(n)}(x^{(n)}) - \hat{Q}(\Omega_n)\hat{F}_Q^{(n)}(x^{(n)})$. Note that $\hat{g}_n \xrightarrow{a.u.} g$ by the Glivenko-Cantelli theorem and $\hat{P} \xrightarrow{a.s.} P$ by the strong law of large numbers.

$$
\begin{aligned}
\left| d_{CM} - \hat{d}_{CM} \right| &= \frac{1}{2} \left| \sum_{n \in \mathbb{N}} \int g_n^2 \mathrm{d}P|_n + \sum_{n \in \mathbb{N}} \int g_n^2 \mathrm{d}Q|_n - \sum_{n \in \mathbb{N}} \sum_i \hat{g}_n(x_i)^2 - \sum_{n \in \mathbb{N}} \sum_i \hat{g}_n(y_i)^2 \right| \\
&= \left| \sum_{n \in \mathbb{N}} \left[ \int g_n^2 \mathrm{d}P|_n - \int \hat{g}_n^2 \mathrm{d}\hat{P}|_n + \int g_n^2 \mathrm{d}Q|_n - \int \hat{g}_n^2 \mathrm{d}\hat{Q}|_n \right] \right| \\
&\leq \left| \sum_{n \in \mathbb{N}} \left[ \left| \int g_n^2 \mathrm{d}P|_n - \int \hat{g}_n^2 \mathrm{d}\hat{P}|_n \right| + \left| \int g_n^2 \mathrm{d}Q|_n - \int \hat{g}_n^2 \mathrm{d}\hat{Q}|_n \right| \right] \right|
\end{aligned}
$$

where $\hat{P} = \sum_i \delta(x_i)$ and $\hat{Q} = \sum_i \delta(y_i)$ are the corresponding empirical measures. Without loss of generality, we only find the bound on $\left| \int g_n^2 \mathrm{d}P|_n - \int \hat{g}_n^2 \mathrm{d}\hat{P}|_n \right|$, then the rest is bounded similarly for $Q$.

$$
\begin{aligned}
\left| \int g_n^2 \mathrm{d}P|_n - \int \hat{g}_n^2 \mathrm{d}\hat{P}|_n \right| &= \left| \int g_n^2 \mathrm{d}P|_n - \int \hat{g}_n^2 \mathrm{d}P|_n + \int \hat{g}_n^2 \mathrm{d}P|_n - \int \hat{g}_n^2 \mathrm{d}\hat{P}|_n \right| \\
&\leq \left\| \left| \int \left( g_n^2 - \hat{g}_n^2 \right) \mathrm{d}P|_n \right| - \left| \int \hat{g}_n^2 \mathrm{d}\left( P|_n - \hat{P}|_n \right) \right| \right\|
\end{aligned}
$$

Applying Glivenko-Cantelli theorem and strong law of large numbers, these two terms converges since $\hat{g}_n^2$ is bounded. Hence, we show that the C-M test estimator is consistent. □

## 5  Results

We present a set of two-sample problems and apply various statistics to perform hypothesis testing. As a baseline measure, we consider the widely used Wilcoxon rank-sum test (or equivalently, the Mann-Whitney U test) on the count distribution (e.g. [9]), which is a non-parametric median test for the total number of action potentials, and the integrated squared deviation statistic $\lambda_{L2} = \int \left( \lambda_1(t) - \lambda_2(t) \right)^2 \mathrm{d}t$, where $\lambda(t)$ is estimated by smoothing spike timing with a Gaussian kernel, evaluated at a uniform grid at least an order of magnitude smaller than the standard deviation of the kernel. We report the performance of the test with varying kernel sizes.

All tests are quantified by the power of the test given a significance threshold (type-I error) at $0.05$. The null hypothesis distribution is empirically computed by either generating independent samples or by permuting the data to create at least 1000 values.

### 5.1  Stationary renewal processes

Renewal process is a widely used point process model that compensates the deviation from Poisson process [10]. We consider two stationary renewal processes with gamma interval distributions. Since the mean rate of the two processes are the same, the rate function statistic and Wilcoxon test does

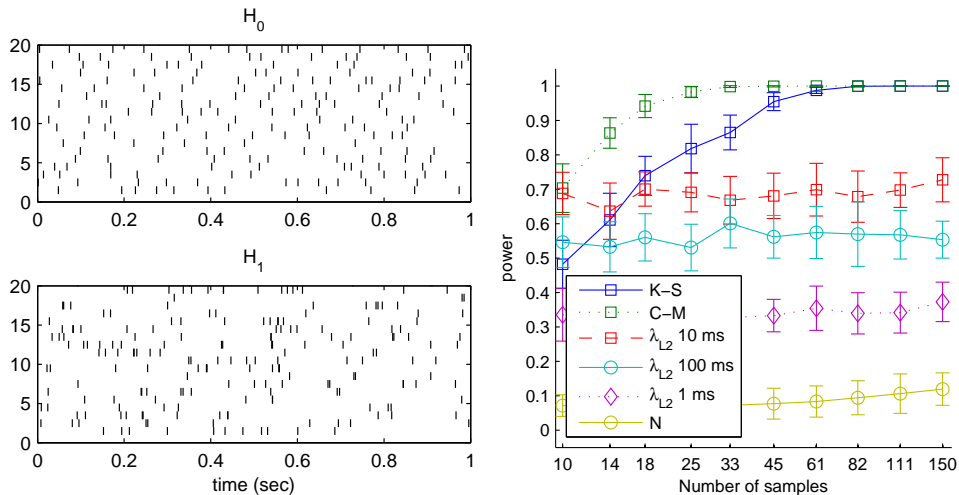

Figure 2: Gamma distributed renewal process with shape parameter $\theta = 3$ ($H_0$) and $\theta = 0.5$ ($H_1$). The mean number of action potential is fixed to 10. (Left) Spike trains from the null and alternate hypothesis. (Right) Comparison of the power of each method. The error bars are standard deviation over 20 Monte Carlo runs.

not yield consistent result, while the proposed measures obtain high power with a small number of samples. The C-M test is more powerful than K-S in this case; this can be interpreted by the fact that the difference in the cumulative is not concentrated but spread out over time because of the stationarity.

## 5.2 Precisely timed spike trains

When the same stimulation is presented to a neuronal system, the observed spike trains sometimes show a highly repeatable spatio-temporal pattern at the millisecond time scale. Recently these precisely timed spike trains (PTST) are abundantly reported both *in vivo* and *in vitro* preparations [11, 12, 13]. Despite being highly reproducible, different forms of trial-to-trial variability have also been observed [14]. It is crucial to understand this variability since for a system to utilize PTSTs as a temporal code, it should presumably be robust to its variability structure, and possibly learn to reduce it [15].

A precisely timed spike train in an interval is modeled by $L$ number of probability density and probability pairs $\{(f_i(t), p_i)\}_{i=1}^{L}$. Each $f_i(t)$ corresponds to the temporal jitter, and $p_i$ corresponds to the probability of generating the spike. Each realization of the PTST model produces at most $L$ spikes. The equi-intensity Poisson process has the rate function $\lambda(t) = \sum_i p_i f_i(t)$. We test if the methods can differentiate between the PTST ($H_0$) and equi-intensity Poisson process ($H_1$) for $L = 1, 2, 3, 4$ (see Figure 3 for the $L = 4$ case). Note that $L$ determines the maximum dimension for the PTST. $f_i(t)$ were equal variance Gaussian distribution on a grid sampled from a uniform random variable, and $p_i = 0.9$.

As shown in Figure 3, only the proposed methods perform well. Since the rate function profile is identical for both models, the rate function statistic $\lambda_{L2}$ fails to differentiate. The Wilcoxon test does work for intermediate dimensions, however its performance is highly variable and unpredictable. In contrast to the previous example, the K-S test is consistently better than the C-M statistic in this problem.

## 6 Optimal stimulation parameter selection

Given a set of point processes, we can find the one which is closest to a target point process in terms of the proposed divergence. Here we use this method on a real dataset obtained from the somatosensory system of an anesthetized rat (see supplement for procedure). Specifically, we address finding

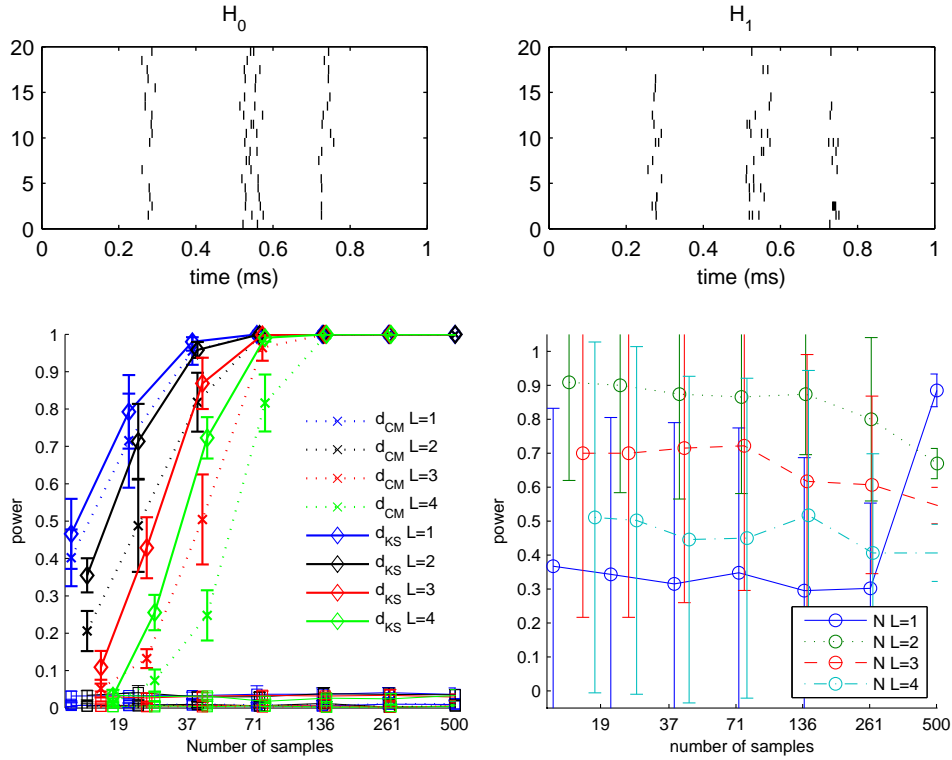

Figure 3: [Top] Precisely timed spike train model ($H_0$) versus equi-intensity Poisson process ($H_1$). Spike trains from the null and alternate hypothesis for $L = 4$. [Bottom] Comparison of the power of each method for $L = 1, 2, 3, 4$ on precisely timed spike train model ($H_0$) versus equi-intensity Poisson process ($H_1$). (Left) Power comparison for methods except for $N$. The rate statistic $\lambda_{L2}$ are not labeled, since they are not able to detect the difference. (Right) Wilcoxon test on the number of action potentials. The error bars are standard deviation over 10 Monte Carlo runs.

optimal electrical stimulation settings to produce cortical spiking patterns similar to those observed with tactile stimuli.

The target process has 240 realizations elicited by tactile stimulation of the ventral side of the first digit with a mechanical tactor. We seek the closest out of 19 processes elicited by electrical stimulation in the thalamus. Each process has 140 realizations that correspond to a particular setting of electrical stimulation. The settings correspond to combinations of duration and amplitude for biphasic current injection on two adjacent channels in the thalamus. The channel of interest and the stimulating channels were chosen to have significant response to tactile stimulation.

The results from applying the C-M, K-S, and $\lambda_{L2}$ measures between the tactile responses and the sets from each electrical stimulation setting are shown Figure 4. The overall trend among the measures is consistent, but the location of the minima does not coincide for $\lambda_{L2}$.

# 7  Conclusion

In this paper, we have proposed two novel measures of divergence between point processes. The proposed measures have been derived from the basic probability law of a point process and we have shown that these measures can be efficiently estimated consistently from data. Using divergences for statistical inference transcends first and second order statistics, and enables distribution-free spike train analysis.

The time complexity of both methods is $\mathcal{O}\left(\sum_n n \left[N_P(n)N_Q(n) + N_P^2(n) + N_Q^2(n)\right]\right)$ where $N_P(n)$ is the number of spike trains from $P$ that has $n$ spikes. In practice this is often faster than

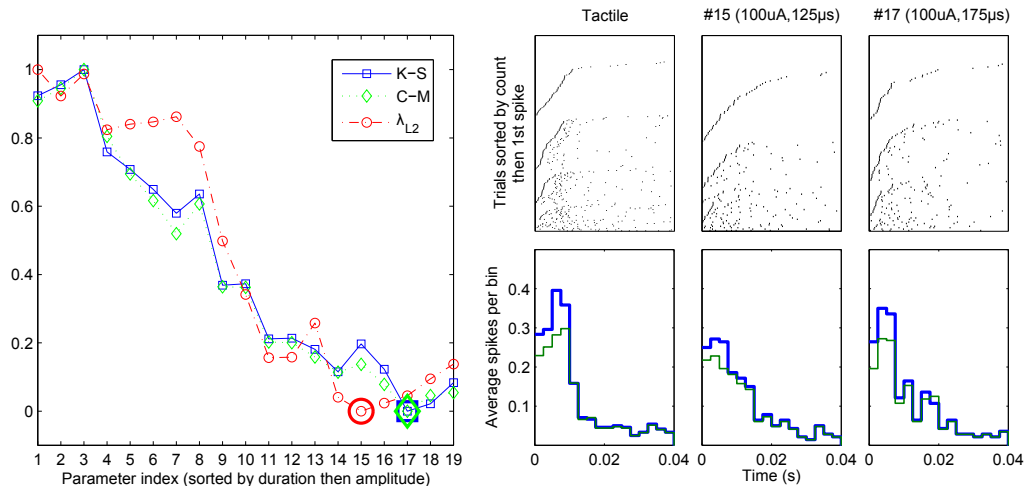

Figure 4: (Left) Dissimilarity/divergences from tactile response across parameter sets. The values of each measure are shifted and scaled to be in the range of 0 to 1. $\lambda_{L2}$ uses 2.5 ms bins with no smoothing. (Right) Responses from the tactile response (left), stimulation settings selected by $\lambda_{L2}$ (center), and the realizations selected by K-S and C-M (right). Top row shows the spike trains stratified into number of spikes and then sorted by spike times. Bottom row shows the average response binned at 2.5 ms; the variance is shown as a thin green line.

the binned rate function estimation which has time complexity $\mathcal{O}(BN)$ where $B$ is the number of bins and $N = \sum_n n(N_P(n) + N_Q(n))$ is the total number of spikes in all the samples. Although, we have observed that the statistic based on the $L_2$ distance between the rate functions often outperforms the proposed method, this approach involves the search for the smoothing kernel size and bin size which can make the process slow and prohibitive. In addition, it brings the danger of multiple testing, since some smoothing kernel sizes may pickup spurious patterns that are only fluctuations due to finite samples size.

A similar approach based on stratification has also been addressed in [16], where the authors have discussed the problem of estimating Hellinger distance between two point processes. Although conceptually similar, the advantage of the proposed approach is that it is parameter free, whereas the other approach requires selecting appropriate kernels and the corresponding kernel sizes for each Euclidean partitions. However, a stratification-based approach suffers in estimation when the count distributions of the point processes under consideration are flat, since in this situation the spike train realizations tend to exist in separate Euclidean partitions, and given a finite set of realizations, it becomes difficult to populate each partition sufficiently. Therefore, other methods should be investigated that allow two spike trains to interact irrespective of their spike counts. Other possible approaches include the kernel-based divergence measures as proposed in [17], since the measures can be applied to any abstract space. However, it requires desinging an appropriate strictly positive definite kernel on the space of spike trains.

In this literature, we have presented the divergences in the context of spike trains generated by neurons. However, the proposed methods can be used for general point processes, and can be applied to other areas. Although we have proved consistency of the proposed measures, further statistical analysis such as small sample power analysis, rate of convergence, and asymptotic properties would be interesting to address. A MATLAB implementation is freely available on the web (http://code.google.com/p/iocane) with BSD-license.

## Acknowledgment

This work is partially funded by NSF Grant ECCS-0856441 and DARPA Contract N66001-10-C-2008.

# References

[1] D. J. Daley and D. Vere-Jones. *An Introduction to the Theory of Point Processes*. Springer, 1988.

[2] D. H. Johnson, C. M. Gruner, K. Baggerly, and C. Seshagiri. Information-theoretic analysis of neural coding. *Journal of Computational Neuroscience*, 10(1):47–69, 2001.

[3] J. D. Victor. Spike train metrics. *Current Opinion in Neurobiology*, 15:585–592, 2005.

[4] A. Kuhn, A. Aertsen, and S. Rotter. Higher-order statistics of input ensembles and the response of simple model neurons. *Neural Computation*, 15(1):67–101, 2003.

[5] F. Rieke, D. Warland, R. de Ruyter van Steveninck, and W. Bialek. *Spikes: exploring the neural code*. MIT Press, Cambridge, MA, USA, 1999.

[6] B. W. Silverman. *Density Estimation for Statistics and Data Analysis*. Chapman and Hall, New York, 1986.

[7] G. Fasano and A. Franceschini. A multidimensional version of the Kolmogorov–Smirnov test. *Royal Astronomical Society, Monthly Notices*, 225:155–170, 1987.

[8] T. W. Anderson. On the distribution of the two-sample Cramér–von-Mises criterion. *Annals of Mathematical Statistics*, 33(3):1148–1159, 1962.

[9] A. Kepecs, N. Uchida, H. A. Zariwala, and Z. F. Mainen. Neural correlates, computation and behavioural impact of decision confidence. *Nature*, 455(7210):227–231, 2008.

[10] M. P. P. Nawrot, C. Boucsein, V. R. Molina, A. Riehle, A. Aertsen, and S. Rotter. Measurement of variability dynamics in cortical spike trains. *Journal of Neuroscience Methods*, 169(2):374–390, 2008.

[11] P. Reinagel and R. Clay Reid. Precise firing events are conserved across neurons. *Journal of Neuroscience*, 22(16):6837–6841, 2002.

[12] M. R. DeWeese, M. Wehr, and A. M. Zador. Binary spiking in auditory cortex. *Journal of Neuroscience*, 23(21):7940–7949, 2003.

[13] R. S. Johansson and I. Birznieks. First spikes in ensembles of human tactile afferents code complex spatial fingertip events. *Nature Neuroscience*, 7(2):170–177, 2004.

[14] P. Tiesinga, J. M. Fellous, and T. J. Sejnowski. Regulation of spike timing in visual cortical circuits. *Nature Reviews Neuroscience*, 9:97–107, 2008.

[15] S. M. Bohte and M. C. Mozer. Reducing the variability of neural responses: A computational theory of spike-timing-dependent plasticity. *Neural Computation*, 19(2):371–403, 2007.

[16] I. Park and J. C. Príncipe. Quantification of inter-trial non-stationarity in spike trains from periodically stimulated neural cultures. In *Proceedings of IEEE International Conference on Acoustics, Speech and Signal Processing*, 2010. Special session on Multivariate Analysis of Brain Signals: Methods and Applications.

[17] A. Gretton, K. M. Borgwardt, M. J. Rasch, B. Schölkopf, and A. J. Smola. A kernel method for the two-sample problem. *CoRR*, abs/0805.2368, 2008.

